# Functional network reorganization in motor cortex can be explained by reward-modulated Hebbian learning

**Robert Legenstein**[1]*, **Steven M. Chase**[2,3,4], **Andrew B. Schwartz**[2,3], **Wolfgang Maass**[1]
[1] Institute for Theoretical Computer Science, Graz University of Technology, Austria
[2] Department of Neurobiology, University of Pittsburgh
[3] Center for the Neural Basis of Cognition
[4] Department of Statistics, Carnegie Mellon University

## Abstract

The control of neuroprosthetic devices from the activity of motor cortex neurons benefits from learning effects where the function of these neurons is adapted to the control task. It was recently shown that tuning properties of neurons in monkey motor cortex are adapted selectively in order to compensate for an erroneous interpretation of their activity. In particular, it was shown that the tuning curves of those neurons whose preferred directions had been misinterpreted changed more than those of other neurons. In this article, we show that the experimentally observed self-tuning properties of the system can be explained on the basis of a simple learning rule. This learning rule utilizes neuronal noise for exploration and performs Hebbian weight updates that are modulated by a global reward signal. In contrast to most previously proposed reward-modulated Hebbian learning rules, this rule does not require extraneous knowledge about what is noise and what is signal. The learning rule is able to optimize the performance of the model system within biologically realistic periods of time and under high noise levels. When the neuronal noise is fitted to experimental data, the model produces learning effects similar to those found in monkey experiments.

## 1  Introduction

It is a commonly accepted hypothesis that adaptation of behavior results from changes in synaptic efficacies in the nervous system. However, there exists little knowledge about how changes in synaptic efficacies change behavior and about the learning principles that underlie such changes. Recently, one important hint has been provided in the experimental study [1] of a monkey controlling a neuroprostethic device. The monkey's intended movement velocity vector can be extracted from the firing rates of a group of recorded units by the population vector algorithm, i.e., by computing the weighted sum of their PDs, where each weight is the unit's normalized firing rate [2].[1] In [1], this velocity vector was used to control a cursor in a 3D virtual reality environment. The task for the monkey was to move the cursor from the center of an imaginary cube to a target appearing at one of its corners. It is well known that performance increases with practice when monkeys are trained to move to targets in similar experimental setups, i.e., the function of recorded neurons is adapted such that control over the new artificial "limb" is improved [3]. In [1], it was systematically studied how such reorganization changes the tuning properties of recorded neurons. The authors manipulated the interpretation of recorded firing rates by the readout system (i.e., the system that converts firing

rates of recorded neurons into cursor movements). When the interpretation was altered for a subset of neurons, the tuning properties of the neurons in this subset changed significantly stronger than those of neurons for which the interpretation of the readout system was not changed. Hence, the experiment showed that motor cortical neurons can change their activity specifically and selectively to compensate for an altered interpretation of their activity within some task. Such adjustment strategy is quite surprising, since it is not clear how the cortical adaption mechanism is able to determine for which subset of neurons the interpretation was altered. We refer to this learning effect as the "credit assignment" effect.

In this article, we propose a simple synaptic learning rule and apply it to a model neural network. This learning rule is capable of optimizing performance in a 3D reaching task and it can explain the learning effects described in [1]. It is biologically realistic since weight changes are based exclusively on local variables and a global scalar reward signal $R(t)$. The learning rule is reward-modulated Hebbian in the following sense: Weight changes at synapses are driven by the correlation between a global reward signal, the presynaptic activity, and the difference of the postsynaptic potential from its recent mean (see [4] for a similar approach). Several reward-modulated Hebbian learning rules have been studied for quite some time both in the context of rate-based [5, 6, 7, 8, 4] and spiking models [9, 10, 11, 12, 13, 14, 15, 16]. They turn out to be viable learning mechanisms in many contexts and constitute a biologically plausible alternative [17, 18] to backpropagation based mechanisms preferentially used in artificial neural networks. One important feature of the learning rule proposed in this article is that noisy neuronal output is used for exploration to improve performance. It was often hypothesized that neuronal variability can optimize motor performance. For example in songbirds, syllable variability results in part from variations in the motor command, i. e. the variability of neuronal activity [19]. Furthermore, there exists evidence for the songbird system that motor variability reflects meaningful motor exploration that can support continuous learning [20]. We show that relatively high amounts of noise are beneficial for the adaptation process but not problematic for the readout system. We find that under realistic noise conditions, the learning rule produces effects surprisingly similar to those found in the experiments of [1]. Furthermore, the version of the reward-modulated Hebbian learning rule that we propose does not require extraneous information about what is noise and what is signal. Thus, we show in this study that reward-modulated learning is a possible explanation for experimental results about neuronal tuning changes in monkey pre-motor cortex. This suggests that reward-modulated learning is an important plasticity mechanism for the acquisition of goal-directed behavior.

## 2  Learning effects in monkey motor cortex

In this section, we briefly describe the experimental results of [1] as well as the network that we used to model learning in motor cortex. Neurons in motor and premotor cortex of primates are broadly tuned to intended arm movement direction [21, 3].[2] This sets the basis for the ability to extract intended arm movement from recorded neuronal activity in in these areas. The tuning curve of a direction tuned neuron is given by its firing rate as a function of movement direction. This curve can be fitted reasonably well by a cosine function. The preferred direction (PD) $\mathbf{p}_i \in \mathbb{R}^3$ of a neuron $i$ is defined as the direction in which the cosine fit to its firing rate is maximal, and the modulation depth is defined as the difference in firing rate between the maximum of the cosine fit and the baseline (mean). The experiments in [1] consisted of a sequence of four brain control sessions: *Calibration, Control, Perturbation,* and *Washout.* The tuning functions of an average of 40 recorded neurons were obtained in the *Calibration* session where the monkey moved its hand in a center out reaching task. Those PDs (or manipulated versions of them) were later used for decoding neural trajectories. We refer to PDs used for decoding as "decoding PDs" (dPDs) in order to distinguish them from measured PDs. In *Control*, *Perturbation*, and *Washout* sessions the monkey had to perform a cursor control task in a 3D virtual reality environment (see Figure 1B). The cursor was initially positioned in the center of an imaginary cube, a target position on one of the corners of the cube was randomly selected and made visible. When the monkey managed to hit the target position with the cursor or a 3s time period expired, the cursor position was reset to the origin and a new target position was randomly selected from the eight corners of the imaginary cube. In the *Control* session, the measured PDs were used as dPDs for cursor control. In the *Perturbation* session, the dPDs of a randomly selected subset of neurons (25% or 50% of the recorded neurons) were altered. This was

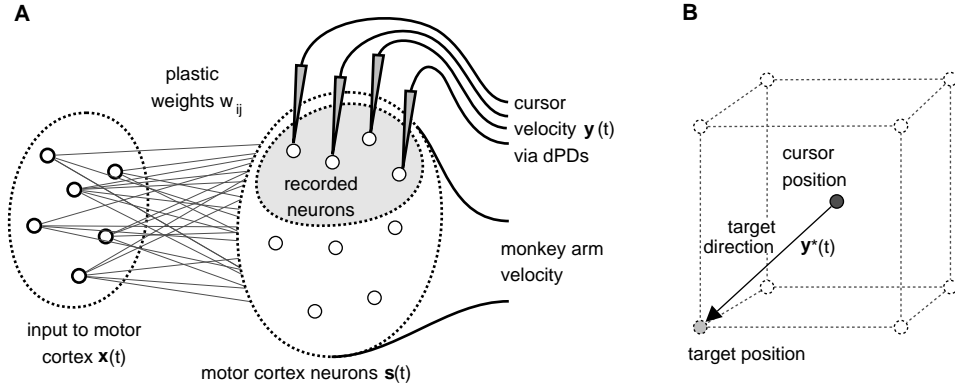

Figure 1: Description of the 3D cursor control task and network model for cursor control. A) Schematic of the network model. A set of $m$ neurons project to $n^{total}$ noisy neurons in motor cortex. The monkey arm movement was modeled by a fixed linear mapping from the activities of the modeled motor cortex neurons to the 3D velocity vector of the monkey arm. A subset of $n$ neurons in the simulated motor cortex was recorded for cursor control. The cursor velocity was given by the population vector. B) The task was to move the cursor from the center of an imaginary cube to one of its eight corners.

achieved by rotating the measured PDs by 90 degrees around the $x$, $y$, or $z$ axes (all PDs were rotated around a single common axis in each experiment). We term these neurons *rotated* neurons. Other dPDs remained the same as in the *Control* session (*non-rotated* neurons). The measured PDs were used for cursor control in the subsequent *Washout* session. In the *Perturbation* session, neurons adapted their firing behavior to compensate for the altered dPDs. The authors observed differential effects of learning for the two groups of non-rotated neurons and rotated neurons. Rotated neurons tended to shift their PDs in the direction of dPD rotation, thus compensating for the perturbation. For non-rotated neurons, the change of the preferred directions was weaker and significantly less strongly biased towards the rotation direction. We refer to this differential behavior of rotated and non-rotated neurons as the "credit assignment effect".

**Network and neuron model:** Our aim in this article is to explain the described effects in the simplest possible model. The model consisted of two populations of neurons, see Figure 1A. The input population modeled those neurons which provide input to the neurons in motor cortex. It consisted of $m = 100$ neurons with activities $x_1(t), \dots, x_m(t) \in \mathbb{R}$. Another population modeled neurons in motor cortex which receive inputs from the input population. It consisted of $n^{total} = 340$ neurons with activities $s_1(t), \dots, s_{n^{total}}(t)$.[3] All modeled motor cortex neurons were used to determine the monkey arm movement in our model. A small number of them ($n = 40$) modeled recorded neurons used for cursor control. We denote the activities of this subset as $s_1(t), \dots, s_n(t)$.

The total synaptic input $a_i(t)$ for neuron $i$ at time $t$ was modeled as a noisy weighted sum of its inputs:

$$a_i(t) = \sum_{j=1}^{m} w_{ij} x_j(t) + \xi_i(t), \qquad \xi_i(t) \text{ drawn from distribution } \mathcal{D}(\nu), \qquad (1)$$

where $w_{ij}$ is the synaptic efficacy from input neuron $j$ to neuron $i$. These weights were set randomly from a uniform distribution in the interval $[-0.5, 0.5]$ at the beginning of each simulation. $\xi_i(t)$ models some exploratory signal needed to explore possibly better network behaviors. In cortical neurons, this exploratory signal could for example result from neuronal or synpatic noise, or it could be spontaneous activity of the neuron. An independent sample from the zero mean distribution $\mathcal{D}(\nu)$ was drawn as the exploratory signal $\xi_i(t)$ at each time step. The parameter $\nu$ (*exploration level*)

determines the variance of the distribution and hence the amount of noise in the neuron. A nonlinear function was applied to the total synaptic input, $s_i(t) = \sigma(a_i(t))$, to obtain the activity $s_i(t)$ of neuron $i$ at time $t$. We used $\sigma : \mathbb{R} \to \mathbb{R}$ is the piecewise linear activation function $\sigma(x) = \max\{x, 0\}$ in order to guarantee non-negative firing rates.

**Task model:** We modeled the cursor control task as shown in Figure 1B. Eight possible cursor target positions were located at the corners of a unit cube in 3D space which had its center at the origin of the coordinate system. At each time step $t$ the desired direction of cursor movement $\mathbf{y}^*(t)$ was computed from the current cursor and target position. By convention, the desired direction $\mathbf{y}^*(t)$ had unit Euclidean norm. From the desired movement direction $\mathbf{y}^*(t)$, the activities $x_1(t), \ldots, x_m(t)$ of the neurons that provide input to the motor cortex neurons were computed and the activities $s_1(t), \ldots, s_n(t)$ of the recorded neurons were used to determine the cursor velocity via their population activity vector (see below).

In order to model the cursor control experiment, we had to determine the PDs of recorded neurons. Obviously, to determine PDs, one needs a model for monkey arm movement. In monkeys, the transformation from motor cortical activity to arm movements involves a complicated system of several synaptic stages. In our model, we treated this transformation as a black box. Experimental findings suggest that monkey arm movements can be predicted quite well by a linear model based on the activities of a small number of motor cortex neurons [3]. We therefore assumed that the direction of the monkey arm movement $\mathbf{y}^{arm}(t)$ at time $t$ can be modeled in a linear way, using the activities of the total population of the $n^{total}$ cortical neurons $s_1(t), \ldots, s_{n^{total}}(t)$ in our simple model and a fixed randomly chosen $3 \times n^{total}$ linear mapping $Q$ (see [23]). With the transformation from motor cortex neurons to monkey arm movements being defined, the input to the network for a given desired direction $\mathbf{y}^*$ should be chosen such that motor cortex neurons produce a monkey arm movement close to the desired movement direction. We therefore calculated from the desired movement direction input activities $\mathbf{x}(t) = c_{rate}(W^{total})^\dagger Q^\dagger \mathbf{y}^*(t)$, where $Q^\dagger$ denotes the pseudo-inverse of $Q$, $W^{total}$ denotes the matrix of weights $w_{ij}$ before learning, and $c_{rate}$ scales the input activity such that the activities of the neurons in the simulated motor cortex could directly be interpreted as rates in Hz [23]. This transformation from desired directions to input neuron activities was defined initially and held fixed during each simulation because learning took place in our model in a single synaptic stage from neurons of the input population to neurons in the motor cortex population in our model and therefore the coding of desired directions did not change in the input population.

As described above, a subset of the motor cortex population was chosen to model recorded neurons that were used for cursor control. For each modeled recorded neuron $i \in \{1, \ldots, n\}$, we determined the preferred direction $\mathbf{p}_i \in \mathbb{R}^3$ as well as the baseline activity $\beta_i$ and the modulation depth $\alpha_i$ by fitting a cosine tuning on the basis of simulated monkey arm movements [1, 23]. In the simulation of a *Perturbation* session, dPDs $\tilde{\mathbf{p}}_i$ of rotated neurons were rotated versions of the measured PDs $\mathbf{p}_i$ (as in [1], one of the $x$, $y$, or $z$ axis was chosen and the PDs were rotated by 90 degrees around this axis), whereas the dPDs of non-rotated neurons were identical to their measured PDs. The dPDs were then used to determine the movement velocity $\mathbf{y}(t)$ of the cursor by the population vector algorithm [1, 2, 23]. This decoding strategy is consistent with an interpretation of the neural activity which codes for the velocity of the movement.

## 3  Adaptation with an online learning rule

Adaptation of synaptic efficacies $w_{ij}$ from input neurons to neurons in motor cortex is necessary if the actual decoding PDs $\tilde{\mathbf{p}}_i$ do not produce optimal cursor trajectories. Assume that suboptimal dPDs $\tilde{\mathbf{p}}_1, \ldots, \tilde{\mathbf{p}}_n$ are used for decoding. Then for some input $\mathbf{x}(t)$, the movement of the cursor is not in the desired direction $\mathbf{y}^*(t)$. The weights $w_{ij}$ should therefore be adapted such that at every time step $t$ the direction of movement $\mathbf{y}(t)$ is close to the desired direction $\mathbf{y}^*(t)$. We can quantify the angular match $R_{ang}(t)$ at time $t$ by the cosine of the angle between movement direction $\mathbf{y}(t)$ and desired direction $\mathbf{y}^*(t)$: $R_{ang}(t) = \frac{\mathbf{y}(t)^T \mathbf{y}^*(t)}{||\mathbf{y}(t)|| \cdot ||\mathbf{y}^*(t)||}$. This measure has a value of 1 if the cursor moves exactly in the desired direction, it is 0 if the cursor moves perpendicular to the desired direction, and it is -1 if the cursor movement is in the opposite direction.

We assume in our model that all synapses receive information about a global reward $R(t)$. The general idea that a neuromodulatory signal gates local synaptic plasticity was studied in [4]. In that

study, the idea was implemented by learning rules where the weight changes are proportional to the covariance between the reward signal $R$ and some measure of neuronal activity $N$ at the synapse. Here, $N$ could correspond to the presynaptic activity, the postsynaptic activity, or the product of both. The authors showed that such learning rules can explain a phenomenon called Herrnstein's matching law. Interestingly, for the analysis in [4] the specific implementation of this correlation based adaptation mechanism is not important. We investigate in this article a learning rule of this type:

$$\text{EH rule:} \qquad \Delta w_{ij}(t) = \eta \, x_j(t) \left[a_i(t) - \bar{a}_i(t)\right] \left[R(t) - \bar{R}(t)\right], \qquad (2)$$

where $\bar{a}_i(t)$ and $\bar{R}(t)$ denote the low-pass filtered version of $a_i(t)$ and $R(t)$ with an exponential kernel[4]. We refer to this rule as the exploratory Hebb rule (EH rule) in this article. The important feature of this learning rule is that apart from variables which are locally available for each neuron $(x_j(t), a_i(t), \bar{a}_i(t))$, only a single scalar signal, $R(t)$, is needed to evaluate performance.[5] The reward signal $R(t)$ is provided by some neural circuit which evaluates performance of the system. In our simulations, we simply used the angular match $R_{ang}(t)$ as this reward signal. Weight updates of the rule are based on correlations between deviations of the reward signal $R(t)$ and the activation $a_i(t)$ from their means. It adjusts weights such that rewards above mean are reinforced. The EH rule (2) approximates gradient ascent on the reward signal by exploring alternatives to the actual behavior with the help of some exploratory signal $\xi(t)$. The deviation of the activation from the recent mean $a_i(t) - \bar{a}_i(t)$ is an estimate of the exploratory term $\xi_i(t)$ at time $t$ if the mean $\bar{a}_i(t)$ is based on neuron activations $\sum_j w_{ij} x_j(t')$ which are similar to the activation $\sum_j w_{ij} x_j(t)$ at time $t$. Here we make use of (1) the fact that weights are changing very slowly and (2) the continuity of the task (inputs $\mathbf{x}$ at successive time points are similar). Then, (2) can be seen as an approximation of

$$\Delta w_{ij}(t) = \eta \, x_j(t)\xi_i(t) \left[R(t) - \bar{R}(t)\right]. \qquad (3)$$

This rule is a typical node-perturbation learning rule [6, 7, 22, 10] which can be shown to approximate gradient ascent, see e.g. [10]. A simple derivation that shows the link between the EH rule (2) and gradient ascent is given in [23].

The EH learning rule differs from other node-perturbation rules in an important aspect. In many node-perturbation learning rules, the noise needs to be accessible to the learning mechanism separately from the output signal. For example, in [6] and [7] binary neurons were used. The weight updates there depend on the probability of the neuron to output 1. In [10] the noise term is directly incorporated in the learning rule. The EH rule does not directly need the noise signal. Instead a temporally filtered version of the neurons activation is used to estimate the noise signal. Obviously, this estimate is only sufficiently accurate if the input to the neuron is temporally stable on small time scales.

## 4 Comparison with experimentally observed learning effects

In this section, we explore the EH rule (2) in a cursor control task that was modeled to closely match the experimental setup in [1]. Each simulated session consisted of a sequence of movements from the center to a target position at one of the corners of the imaginary cube, with online weight updates during the movements. In monkey experiments, perturbation of decoding PDs lead to retuning of PDs with the above described credit assignment effect [1]. In order to obtain biologically plausible values for the noise distribution in our neuron model, the noise in our model was fitted to data from experiments (see [23]). Analysis of the neuronal responses in the experiments showed that the variance of the response for a given desired direction scaled roughly linearly with the mean firing rate of that neuron for this direction. We obtained this behavior with our neuron model with noise that is a mixture of an activation-independent noise source and a noise source where the variance scales linearly with the activation of the neuron. In particular, the noise term $\xi_i(t)$ of neuron $i$ was drawn from the uniform distribution in $[-\nu_i(\mathbf{x}(t)), \nu_i(\mathbf{x}(t))]$ with an exploration level $\nu_i$ given by

$\nu_i(\mathbf{x}(t)) = 10 + 2.8\sqrt{\sigma\left(\sum_{j=1}^{m} w_{ij}x_j(t)\right)}$. The constants where chosen fit neuron behavior in the

data. We note that in all simulations with the EH rule, the input activities $x_j(t)$ were scaled in such a way that the output of the neuron at time $t$ could be interpreted directly as the firing rate of the neuron

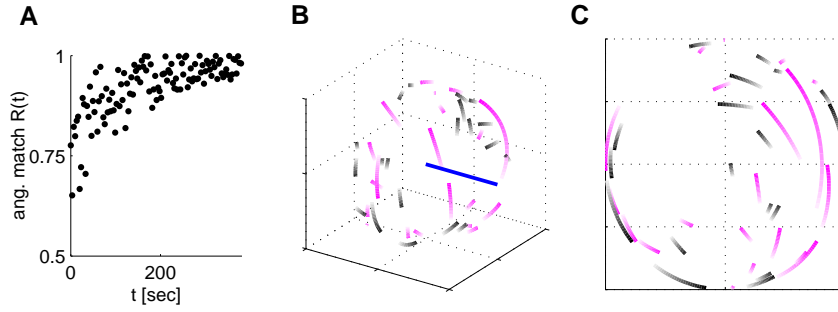

Figure 2: One example simulation of the 50% perturbation experiment with the EH rule and data-derived network parameters. A) Angular match $R_{ang}$ as a function of learning time. Every 100th time point is plotted. B) PD shifts drawn on the unit sphere (arbitrary units) for non-rotated (black traces) and rotated (light cyan traces) neurons from their initial values (light) to their values after training (dark, these PDs are connected by the shortest path on the unit sphere). The straight line indicates the rotation axis. C) Same as B, but the view was altered such that the rotation axis is directed towards the reader. The PDs of rotated neurons are consistently rotated in order to compensate for the perturbation.

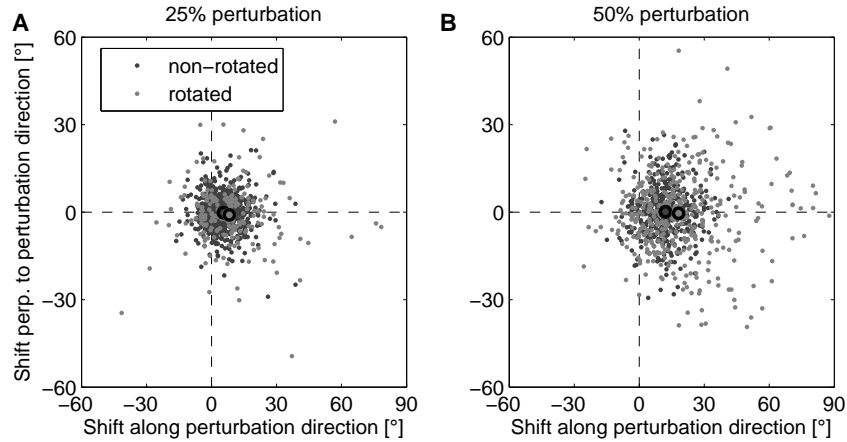

Figure 3: PD shifts in simulated *Perturbation* sessions are in good agreement with experimental results (compare to Figure 3A,B in [1]). Shift in the PDs measured after simulated perturbation sessions relative to initial PDs for all units in 20 simulated experiments where 25% (A) or 50% (B) of the units were rotated. Dots represent individual data points and black circled dots represent the means of rotated (light gray) and non-rotated (dark gray) units.

at time $t$. With such scaling, we obtained output values of the neurons without the exploratory signal in the range of 0 to 120Hz with a roughly exponential distribution. Having estimated the variability of neuronal response, the learning rate $\eta$ remained the last free parameter of the model. To constrain this parameter, $\eta$ was chosen such that the performance in the 25% perturbation task approximately matched the monkey performance.

We simulated the two types of perturbation experiments reported in [1] in our model network with 40 recorded neurons. In the first set of simulations, a random set of 25% of recorded neurons were rotated neurons in *Perturbation* sessions. In the second set of simulations, we chose 50 % of the recorded neurons to be rotated. In each simulation, 320 targets were presented to the model, which is similar to the number of target presentations in [1]. Results for one example run are shown in Figure 2. The shifts in PDs of recorded neurons induced by training in 20 independent trials were compiled and analyzed separately for rotated neurons and non-rotated neurons. The results are in good agreement with the experimental data, see Figure 3. In the simulated 25% perturbation

experiment, the mean shift of the PD for rotated neurons was $8.2 \pm 4.8$ degrees, whereas for non-rotated neurons, it was $5.5 \pm 1.6$ degrees. This relatively small effect is similar to the effect observed in [1] where the PD shift of rotated (non-rotated) units was 9.9 (5.2) degrees. The effect is more pronounced in the 50% perturbation experiment (see below). We also compared the deviation of the movement trajectory from the ideal straight line in rotation direction half way to the target[6] from early trials to the deviation of late trials, where we scaled the results to a cube of 11cm side length in order to be able to compare the results directly to the results in [1]. In early trials, the trajectory deviation was $9.2 \pm 8.8$mm, which was reduced by learning to $2.4 \pm 4.9$mm. In the simulated 50% perturbation experiment, the mean shift of the PD for rotated neurons was $18.1 \pm 4.2$ degrees, whereas for non-rotated neurons, it was $12.1 \pm 2.6$ degrees (in monkey experiments [1] this was 21.7 and 16.1 degrees respectively). The trajectory deviation was $23.1 \pm 7.5$mm in early trials, and $4.8 \pm 5.1$mm in late trials. Here, the early deviation was stronger than in the monkey experiment, while the late deviation was smaller.

The EH rule (2) falls into the general class of correlation-based learning rules described in [4]. In these rules the weight change is proportional to the covariance of the reward signal and some measure of neuronal activity. We performed the same experiment with slightly different correlation-based rules

$$\Delta w_{ij}(t) = \eta \, x_j(t) a_i(t) \left[ R(t) - \bar{R}(t) \right], \tag{4}$$
$$\Delta w_{ij}(t) = \eta \, x_j(t) \left[ a_i(t) - \bar{a}_i(t) \right] R(t), \tag{5}$$

(compare to (2)). The performance improvements were similar to those obtaint with the EH rule. However, no credit assignment effect was observed with these rules. In the simulated 50% perturbation experiment, the mean shift of the PD of rotated neurons (non-rotated neurons) was $12.8 \pm 3.6$ $(12.0 \pm 2.4)$ degrees for rule (4) and $25.5 \pm 4$ $(26.8 \pm 2.8)$ degrees for rule (5).

In the monkey experiment, training in the *Perturbation* session also induced in a decrease of the modulation depth of rotated neurons. This resulted in a decreased contribution of these neurons to the cursor movement. We observed a qualitatively similar resultin our simulations. In the 25% perturbation simulation, modulation depths decreased on average by $2.7 \pm 4.3$Hz for rotated neurons. Modulation depths for non-rotated neurons increased on average by $2.2 \pm 3.9$Hz (average over 20 independent simulations). In the 50% perturbation simulation, the changes in modulation depths were $-3, 6 \pm 5.5$Hz for rotated neurons and $5.4 \pm 6$Hz for non-rotated neurons.[7] Thus, the relative contribution of rotated neurons on cursor movement decreased.

Comparing the results obtained by our simulations to those of monkey experiments (compare Figure 3 to Figure 3 in [1]), it is interesting that quantitatively similar effects were obtained when noise level and learning rate was constrained by the experimental data. One should note here that tuning changes due to learning depend on the noise level. For small exploration levels, PDs changed only slightly and the difference in PD change between rotated and non-rotated neurons was small, while for large noise levels, PD change differences can be quite drastic. Also the learning rate $\eta$ influences the amount of PD shift differences with higher learning rates leading to stronger credit assignment effects, see [23] for details.

The performance of the system before and after learning is shown in Figure 4. The neurons in the network after training are subject to the same amount of noise as the neurons in the network before training, but the angular match after training shows much less fluctuation than before training. Hence, the network automatically suppresses jitter on the trajectory in the presence of high exploration levels $\nu$. We quantified this observation by computing the standard deviation of the angle between the cursor velocity vector and the desired movement direction for 100 randomly drawn noise samples.[8] The mean standard deviation for 50 randomly drawn target directions was always decreased by learning. In the mean over the 20 simulations, the mean STD over 50 target directions was 7.9 degrees before learning and 6.3 degrees after learning. Hence, the network not only adapted its response to the input, it also found a way to optimize its sensitivity to the exploratory signal.

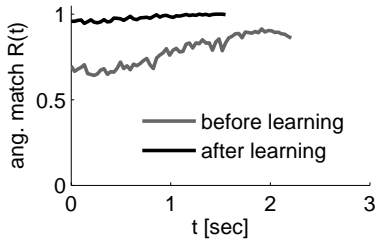

Figure 4: Network performance before and after learning for 50% perturbation. Angular match $R_{ang}(t)$ of the cursor movements in one reaching trial before (gray) and after (black) learning as a function of the time since the target was first made visible. The black curve ends prematurely because the target was reached faster. After learning temporal jitter of the performance was reduced, indicating reduced sensitivity to noise.

## 5 Discussion

Jarosiewicz et al. [1] discussed three strategies that could potentially be used by the monkey to compensate for the errors caused by perturbations: re-aiming, re-weighting, and re-mapping. Using the re-aiming strategy, the monkey compensates for perturbations by aiming for a virtual target located in the direction that offsets the visuomotor rotation. The authors identified a global change in the activity level of all neurons. This indicates a re-aiming strategy of the monkey. Re-weighting would suppress the use of rotated units, leading to a reduction of their modulation depths. A reduction of modulation depths of rotated neurons was also identified in the experimentals. A re-mapping strategy would selectively change the directional tunings of rotated units. Rotated neurons shifted their PDs more than the non-rotated population in the experiments. Hence, the authors found elements of all three strategies in their data. These three elements of neuronal adaptation were also identified in our model: a global change in activity of neurons (all neurons changed their tuning properties; re-aiming), a reduction of modulation depths for rotated neurons (re-weighting), and a selective change of the directional tunings of rotated units (re-mapping). This modeling study therefore suggests that all three elements can be explained by a single synaptic adaptation strategy that relies on noisy neuronal activity and visual feedback that is made accessible to all synapses in the network by a global reward signal. It is noteworthy that the credit assignment phenomenon is an emergent feature of the learning rule rather than implemented in some direct way. Intuitively, this behavior can be explained in the following way. The output of non-rotated neurons is consistent with the interpretation of the readout system. So if this output is strongly altered, performance will likely drop. On the other hand, if the output of a rotated neuron is radically different, this will often improve performance. Hence, the relatively high noise levels measured in experiments are probably important for the credit assignment phenomenon. Under such realistic noise conditions, our model produced effects surprisingly similar to those found in the monkey experiments. Thus, this study shows that reward-modulated learning can explain detailed experimental results about neuronal adaptation in motor cortex and therefore suggests that reward-modulated learning is an essential plasticity mechanism in cortex.

The results of this modeling paper also support the hypotheses introduced in [24]. The authors presented data which suggests that neural representations change randomly (background changes) even without obvious learning, while systematic task-correlated representational changes occur within a learning task.

Reward-modulated Hebbian learning rules are currently the most promising candidate for a learning mechanism that can support goal-directed behavior by local synaptic changes in combination with a global performance signal. The EH rule (2) is one particularly simple instance of such rules that exploits temporal continuity of inputs and an exploration signal - a signal which would show up as "noise" in neuronal recordings. We showed that large exploration levels are beneficial for learning while they do not interfere with the performance of the system because of pooling effects of readout elements. This study therefore provides a hypothesis about the role of "noise" or ongoing activity in cortical circuits as a source for exploration utilized by local learning rules.

**Acknowledgments**

This work was supported by the Austrian Science Fund FWF [S9102-N13, to R.L. and W.M.]; the European Union [FP6-015879 (FACETS), FP7-216593 (SECO), FP7-506778 (PASCAL2), FP7-231267 (ORGANIC) to R.L. and W.M.]; and by the National Institutes of Health [R01-NS050256, EB005847, to A.B.S.].

## Footnotes

*To whom correspondence should be addressed: `robert.legenstein@igi.tugraz.at`

[1] In general, a unit is not necessarily equal to a neuron in the experiments. Since the spikes of a unit are determined by a spike sorting algorithm, a unit may represent the mixed activity of several neurons.

[2]Arm movement refers to movement of the endpoint of the arm.

[3]The distinction between these two layers is purely functional. Input neurons may be situated in extracortical areas, in other cortical areas, or even in motor cortex itself. The functional feature of these two populations in our model is that learning takes place solely in synapses of projections between these population since the aim of this article is to explain the learning effects in the simplest model. But in principle the same learning is applicable to multilayer networks.

[4]We used $\bar{a}_i(t) = 0.8\bar{a}_i(t-1) + 0.2a_i(t)$ and $\bar{R}(t) = 0.8\bar{R}(t-1) + 0.2R(t)$

[5]A rule where the activation $a_i$ is replaced by the output $s_i$ and obtained very similar results.

[6]These deviations were computed as described in [1]

[7]When comparing these results to experimental results, one has to take into account the modulation depths in monkey experiments were around 10Hz whereas in the simulations, they were around 25Hz

[8]This effect is not caused by a larger norm of the weight vectors. The comparison was done with weight vectors after training normalized to their L2 norm before training.

# References

[1] B. Jarosiewicz, S. M. Chase, G. W. Fraser, M. Velliste, R. E. Kass, and A. B. Schwartz. Functional network reorganization during learning in a brain-computer interface paradigm. *Proc. Nat. Acad. Sci. USA*, 105(49):19486–91, 2008.

[2] A. P. Georgopoulos, R. E. Ketner, and A. B. Schwartz. Primate motor cortex and free arm movements to visual targets in three- dimensional space. ii. coding of the direction of movement by a neuronal population. *J. Neurosci.*, 8:2928–2937, 1988.

[3] A. B. Schwartz. Useful signals from motor cortex. *J. Physiology*, 579:581–601, 2007.

[4] Y. Loewenstein and H. S. Seung. Operant matching is a generic outcome of synaptic plasticity based on the covariance between reward and neural activity. *Proc. Nat. Acad. Sci. USA*, 103(41):15224–15229, 2006.

[5] A. G. Barto, R. S. Sutton, and C. W. Anderson. Neuronlike adaptive elements that can solve difficult learning control problems. *IEEE Trans. Syst. Man Cybern.*, SMC-13(5):834–846, 1983.

[6] P. Mazzoni, R. A. Andersen, and M. I. Jordan. A more biologically plausible learning rule for neural networks. *Proc. Nat. Acad. Sci. USA*, 88(10):4433–4437, 1991.

[7] R. J. Williams. Simple statistical gradient-following algorithms for connectionist reinforcement learning. *Machine Learning*, 8:229–256, 1992.

[8] J. Baxter and P. L. Bartlett. Direct gradient-based reinforcement learning: I. gradient estimation algorithms. Technical report, Research School of Information Sciences and Engineering, Australian National University, 1999.

[9] X. Xie and H. S. Seung. Learning in neural networks by reinforcement of irregular spiking. *Phys. Rev. E*, 69(041909), 2004.

[10] I. R. Fiete and H. S. Seung. Gradient learning in spiking neural networks by dynamic perturbation of conductances. *Phys. Rev. Lett.*, 97(4):048104–1 to 048104–4, 2006.

[11] J.-P. Pfister, T. Toyoizumi, D. Barber, and W. Gerstner. Optimal spike-timing-dependent plasticity for precise action potential firing in supervised learning. *Neural Computation*, 18(6):1318–1348, 2006.

[12] E. M. Izhikevich. Solving the distal reward problem through linkage of STDP and dopamine signaling. *Cerebral Cortex*, 17:2443–2452, 2007.

[13] D. Baras and R. Meir. Reinforcement learning, spike-time-dependent plasticity, and the bcm rule. *Neural Computation*, 19(8):2245–2279, 2007.

[14] R. V. Florian. Reinforcement learning through modulation of spike-timing-dependent synaptic plasticity. *Neural Computation*, 6:1468–1502, 2007.

[15] M. A. Farries and A. L. Fairhall. Reinforcement learning with modulated spike timing-dependent synaptic plasticity. *J. Neurophys.*, 98:3648–3665, 2007.

[16] R. Legenstein, D. Pecevski, and W. Maass. A learning theory for reward-modulated spike-timing-dependent plasticity with application to biofeedback. *PLoS Computational Biology*, 4(10):1–27, 2008.

[17] C. H. Bailey, M. Giustetto, Y.-Y. Huang, R. D. Hawkins, and E. R. Kandel. Is heterosynaptic modulation essential for stabilizing Hebbian plasticity and memory? *Nat. Rev. Neurosci.*, 1:11–20, 2000.

[18] Q. Gu. Neuromodulatory transmitter systems in the cortex and their role in cortical plasticity. *Neuroscience*, 111(4):815–835, 2002.

[19] Samuel J. Sober, Melville J. Wohlgemuth, and Michael S. Brainard. Central contributions to acoustic variation in birdsong. *J. Neurosci.*, 28(41):10370–9, 2008.

[20] E. C. Tumer and M. S. Brainard. Performance variability enables adaptive plasticity of 'crystallized' adult birdsong. *Nature*, 250(7173):1240–1244, 2007.

[21] A. P. Georgopoulos, A. P. Schwartz, and R. E. Ketner. Neuronal population coding of movement direction. *Science*, 233:1416–1419, 1986.

[22] J. Baxter and P. L. Bartlett. Infinite-horizon policy-gradient estimation. *J. Artif. Intell. Res.*, 15:319–350, 2001.

[23] R. Legenstein, S. M. Chase, A. B. Schwartz, and W. Maass. A reward-modulated hebbian learning rule can explain experimentally observed network reorganization in a brain control task. *Submitted for publication*, 2009.

[24] U. Rokni, A G. Richardson, E. Bizzi, and H. S. Seung. Motor learning with unstable neural representations. *Neuron*, 54:653–666, 2007.

